# Modeling General and Specific Aspects of Documents with a Probabilistic Topic Model

**Chaitanya Chemudugunta, Padhraic Smyth**
Department of Computer Science
University of California, Irvine
Irvine, CA 92697-3435, USA
{chandra,smyth}@ics.uci.edu

**Mark Steyvers**
Department of Cognitive Sciences
University of California, Irvine
Irvine, CA 92697-5100, USA
msteyver@uci.edu

## Abstract

Techniques such as probabilistic topic models and latent-semantic indexing have been shown to be broadly useful at automatically extracting the topical or semantic content of documents, or more generally for dimension-reduction of sparse count data. These types of models and algorithms can be viewed as generating an abstraction from the words in a document to a lower-dimensional latent variable representation that captures what the document is generally about beyond the specific words it contains. In this paper we propose a new probabilistic model that tempers this approach by representing each document as a combination of (a) a background distribution over common words, (b) a mixture distribution over general topics, and (c) a distribution over words that are treated as being specific to that document. We illustrate how this model can be used for information retrieval by matching documents both at a general topic level and at a specific word level, providing an advantage over techniques that only match documents at a general level (such as topic models or latent-semantic indexing) or that only match documents at the specific word level (such as TF-IDF).

## 1   Introduction and Motivation

Reducing high-dimensional data vectors to robust and interpretable lower-dimensional representations has a long and successful history in data analysis, including recent innovations such as latent semantic indexing (LSI) (Deerwester et al, 1994) and latent Dirichlet allocation (LDA) (Blei, Ng, and Jordan, 2003). These types of techniques have found broad application in modeling of sparse high-dimensional count data such as the "bag of words" representations for documents or transaction data for Web and retail applications.

Approaches such as LSI and LDA have both been shown to be useful for "object matching" in their respective latent spaces. In information retrieval for example, both a query and a set of documents can be represented in the LSI or topic latent spaces, and the documents can be ranked in terms of how well they match the query based on distance or similarity in the latent space. The mapping to latent space represents a generalization or abstraction away from the sparse set of observed words, to a "higher-level" semantic representation in the latent space. These abstractions in principle lead to better generalization on new data compared to inferences carried out directly in the original sparse high-dimensional space. The capability of these models to provide improved generalization has been demonstrated empirically in a number of studies (e.g., Deerwester et al 1994; Hofmann 1999; Canny 2004; Buntine et al, 2005).

However, while this type of generalization is broadly useful in terms of inference and prediction, there are situations where one can over-generalize. Consider trying to match the following query to a historical archive of news articles: **election** + **campaign** + **Camejo**. The query is intended to find documents that are about US presidential campaigns and also about Peter Camejo (who ran as

vice-presidential candidate alongside independent Ralph Nader in 2004). LSI and topic models are likely to highly rank articles that are related to presidential elections (even if they don't necessarily contain the words **election** or **campaign**).

However, a potential problem is that the documents that are highly ranked by LSI or topic models need not include any mention of the name **Camejo**. The reason is that the combination of words in this query is likely to activate one or more latent variables related to the concept of presidential campaigns. However, once this generalization is made the model has "lost" the information about the specific word **Camejo** and it will only show up in highly ranked documents if this word happens to frequently occur in these topics (unlikely in this case given that this candidate received relatively little media coverage compared to the coverage given to the candidates from the two main parties). But from the viewpoint of the original query, our preference would be to get documents that are about the *general topic* of US presidential elections with the *specific constraint* that they mention Peter Camejo.

Word-based retrieval techniques, such as the widely-used term-frequency inverse-document-frequency (TF-IDF) method, have the opposite problem in general. They tend to be overly specific in terms of matching words in the query to documents.

In general of course one would like to have a balance between generality and specificity. One ad hoc approach is to combine scores from a general method such as LSI with those from a more specific method such as TF-IDF in some manner, and indeed this technique has been proposed in information retrieval (Vogt and Cottrell, 1999). Similarly, in the ad hoc LDA approach (Wei and Croft, 2006), the LDA model is linearly combined with document-specific word distributions to capture both general as well as specific information in documents. However, neither method is entirely satisfactory since it is not clear how to trade-off generality and specificity in a principled way.

The contribution of this paper is a new graphical model based on latent topics that handles the trade-off between generality and specificity in a fully probabilistic and automated manner. The model, which we call the special words with background (SWB) model, is an extension of the LDA model. The new model allows words in documents to be modeled as either originating from general topics, or from document-specific "special" word distributions, or from a corpus-wide background distribution. The idea is that words in a document such as **election** and **campaign** are likely to come from a general topic on presidential elections, whereas a name such as **Camejo** is much more likely to be treated as "non-topical" and specific to that document. Words in queries are automatically interpreted (in a probabilistic manner) as either being topical or special, in the context of each document, allowing for a data-driven document-specific trade-off between the benefits of topic-based abstraction and specific word matching. Daumé and Marcu (2006) independently proposed a probabilistic model using similar concepts for handling different training and test distributions in classification problems.

Although we have focused primarily on documents in information retrieval in the discussion above, the model we propose can in principle be used on any large sparse matrix of count data. For example, transaction data sets where rows are individuals and columns correspond to items purchased or Web sites visited are ideally suited to this approach. The latent topics can capture broad patterns of population behavior and the "special word distributions" can capture the idiosyncracies of specific individuals.

Section 2 reviews the basic principles of the LDA model and introduces the new SWB model. Section 3 illustrates how the model works in practice using examples from New York Times news articles. In Section 4 we describe a number of experiments with 4 different document sets, including perplexity experiments and information retrieval experiments, illustrating the trade-offs between generalization and specificity for different models. Section 5 contains a brief discussion and concluding comments.

## 2 A Topic Model for Special Words

Figure 1(a) shows the graphical model for what we will refer to as the "standard topic model" or LDA. There are $D$ documents and document $d$ has $N_d$ words. $\alpha$ and $\beta$ are fixed parameters of symmetric Dirichlet priors for the $D$ document-topic multinomials represented by $\theta$ and the $T$ topic-word multinomials represented by $\phi$. In the generative model, for each document $d$, the $N_d$ words

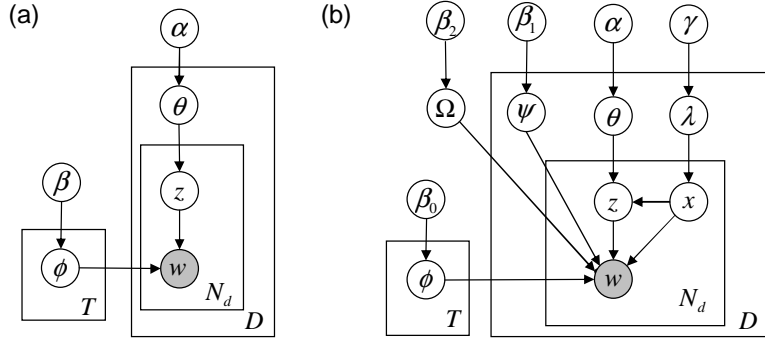

Figure 1: Graphical models for (a) the standard LDA topic model (left) and (b) the proposed special words topic model with a background distribution (SWB) (right).

are generated by drawing a topic $t$ from the document-topic distribution $p(z|\theta_d)$ and then drawing a word $w$ from the topic-word distribution $p(w|z = t, \phi_t)$. As shown in Griffiths and Steyvers (2004) the topic assignments $z$ for each word token in the corpus can be efficiently sampled via Gibbs sampling (after marginalizing over $\theta$ and $\phi$). Point estimates for the $\theta$ and $\phi$ distributions can be computed conditioned on a particular sample, and predictive distributions can be obtained by averaging over multiple samples.

We will refer to the proposed model as the special words topic model with background distribution (SWB) (Figure 1(b)). SWB has a similar general structure to the LDA model (Figure 1(a)) but with additional machinery to handle special words and background words. In particular, associated with each word token is a latent random variable $x$, taking value $x = 0$ if the word $w$ is generated via the topic route, value $x = 1$ if the word is generated as a special word (for that document) and value $x = 2$ if the word is generated from a background distribution specific for the corpus. The variable $x$ acts as a switch: if $x = 0$, the previously described standard topic mechanism is used to generate the word, whereas if $x = 1$ or $x = 2$, words are sampled from a document-specific multinomial $\Psi$ or a corpus specific multinomial $\Omega$ (with symmetric Dirichlet priors parametrized by $\beta_1$ and $\beta_2$) respectively. $x$ is sampled from a document-specific multinomial $\lambda$, which in turn has a symmetric Dirichlet prior, $\gamma$. One could also use a hierarchical Bayesian approach to introduce another level of uncertainty about the Dirichlet priors (e.g., see Blei, Ng, and Jordan, 2003)—we have not investigated this option, primarily for computational reasons. In all our experiments, we set $\alpha = 0.1$, $\beta_0 = \beta_2 = 0.01$, $\beta_1 = 0.0001$ and $\gamma = 0.3$—all weak symmetric priors.

The conditional probability of a word $w$ given a document $d$ can be written as:

$$p(w|d) = p(x = 0|d) \sum_{t=1}^{T} p(w|z = t)p(z = t|d) \; + \; p(x = 1|d)p'(w|d) \; + \; p(x = 2|d)p''(w)$$

where $p'(w|d)$ is the special word distribution for document $d$, and $p''(w)$ is the background word distribution for the corpus. Note that when compared to the standard topic model the SWB model can explain words in three different ways, via topics, via a special word distribution, or via a background word distribution. Given the graphical model above, it is relatively straightforward to derive Gibbs sampling equations that allow joint sampling of the $z_i$ and $x_i$ latent variables for each word token $w_i$, for $x_i = 0$:

$$p\left(x_i = 0, z_i = t \,|\mathbf{w}, \mathbf{x}_{-i}, \mathbf{z}_{-i}, \alpha, \beta_0, \gamma\right) \; \propto \; \frac{N_{d0,-i} + \gamma}{N_{d,-i} + 3\gamma} \times \frac{C_{td,-i}^{TD} + \alpha}{\sum_{t'} C_{t'd,-i}^{TD} + T\alpha} \times \frac{C_{wt,-i}^{WT} + \beta_0}{\sum_{w'} C_{w't,-i}^{WT} + W\beta_0}$$

and for $x_i = 1$:

$$p\left(x_i = 1 \,|\mathbf{w}, \mathbf{x}_{-i}, \mathbf{z}_{-i}, \beta_1, \gamma\right) \; \propto \; \frac{N_{d1,-i} + \gamma}{N_{d,-i} + 3\gamma} \times \frac{C_{wd,-i}^{WD} + \beta_1}{\sum_{w'} C_{w'd,-i}^{WD} + W\beta_1}$$

e mail krugman nytimes com memo to critics of the media s liberal bias the pinkos you really should be going after are those business reporters even i was startled by the tone of the jan 21 issue of investment news which describes itself as the weekly newspaper for financial advisers the headline was paul o neill s sweet deal the blurb was irs backs off closing loophole averting tax liability for execs and treasury chief it s not really news that the bush administration likes tax breaks for businessmen but two weeks later i learned from the wall street journal that this loophole is more than a tax break for businessmen it s a gift to biznesmen and it may be part of a larger pattern confused in the former soviet union the term biznesmen pronounced beeznessmen refers to the class of sudden new rich who emerged after the fall of communism and who generally got rich by using their connections to strip away the assets of public enterprises what we ve learned from enron and other players to be named later is that america has its own biznesmen and that we need to watch out for policies that make it easier for them to ply their trade it turns out that the sweet deal investment news was referring to the use of split premium life insurance policies to give executives largely tax free compensation you don t want to know the details is an even sweeter deal for executives of companies that go belly up it shields their wealth from creditors and even from lawsuits sure enough reports the wall street journal former enron c e o s kenneth lay and jeffrey skilling both had large split premium policies so what other pro biznes policies have been promulgated lately last year both houses of …

john w snow was paid more than 50 million in salary bonus and stock in his nearly 12 years as chairman of the csx corporation the railroad company during that period the company s profits fell and its stock rose a bit more than half as much as that of the average big company mr snow s compensation amid csx s uneven performance has drawn criticism from union officials and some corporate governance specialists in 2000 for example after the stock had plunged csx decided to reverse a 25 million loan to him the move is likely to get more scrutiny after yesterday s announcement that mr snow has been chosen by president bush to replace paul o neill as the treasury secretary like mr o neill mr snow is an outsider on wall street but an insider in corporate america with long experience running an industrial company some wall street analysts who follow csx said yesterday that mr snow had ably led the company through a difficult period in the railroad industry and would make a good treasury secretary it s an excellent nomination said jill evans an analyst at j p morgan who has a neutral rating on csx stock i think john s a great person for the administration he as the c e o of a railroad has probably touched every sector of the economy union officials are less complimentary of mr snow s performance at csx last year the a f l c i o criticized him and csx for the company s decision to reverse the loan allowing him to return stock he had purchased with the borrowed money at a time when independent directors are in demand a corporate governance specialist said recently that mr snow had more business relationships with members of his own board than any other chief executive in addition mr snow is the third highest paid of 37 chief executives of transportation companies said ric marshall chief executive of the corporate library which provides specialized investment research into corporate boards his own compensation levels have been pretty high mr marshall said he could afford to take a public service job a csx program in 1996 allowed mr snow and other top csx executives to buy…

Figure 2: Examples of two news articles with special words (as inferred by the model) shaded in gray. (a) upper, email article with several colloquialisms, (b) lower, article about CSX corporation.

and for $x_i = 2$:

$$p\left(x_i = 2 \,|\, \mathbf{w}, \mathbf{x}_{-i}, \mathbf{z}_{-i}, \beta_2, \gamma\right) \;\propto\; \frac{N_{d2,-i} + \gamma}{N_{d,-i} + 3\gamma} \times \frac{C^W_{w,-i} + \beta_2}{\sum_{w'} C^W_{w',-i} + W\beta_2}$$

where the subscript $-i$ indicates that the count for word token $i$ is removed, $N_d$ is the number of words in document $d$ and $N_{d0}$, $N_{d1}$ and $N_{d2}$ are the number of words in document $d$ assigned to the latent topics, special words and background component, respectively, $C^{WT}_{wt}$, $C^{WD}_{wd}$ and $C^W_w$ are the number of times word $w$ is assigned to topic $t$, to the special-words distribution of document $d$, and to the background distribution, respectively, and $W$ is the number of unique words in the corpus. Note that when there is not strong supporting evidence for $x_i = 0$ (i.e., the conditional probability of this event is low), then the probability of the word being generated by the special words route, $x_i = 1$, or background route, $x_i = 2$ increases.

One iteration of the Gibbs sampler corresponds to a sampling pass through all word tokens in the corpus. In practice we have found that around 500 iterations are often sufficient for the in-sample perplexity (or log-likelihood) and the topic distributions to stabilize.

We also pursued a variant of SWB, the special words (SW) model that excludes the background distribution $\Omega$ and has a symmetric Beta prior, $\gamma$, on $\lambda$ (which in SW is a document-specific Bernoulli distribution). In all our SW model runs, we set $\gamma = 0.5$ resulting in a weak symmetric prior that is equivalent to adding one pseudo-word to each document. Experimental results (not shown) indicate that the final word-topic assignments are not sensitive to either the value of the prior or the initial assignments to the latent variables, $x$ and $z$.

## 3 Illustrative Examples

We illustrate the operation of the SW model with a data set consisting of 3104 articles from the New York Times (NYT) with a total of 1,399,488 word tokens. This small set of NYT articles was formed by selecting all NYT articles that mention the word "Enron." The SW topic model was run with $T = 100$ topics. In total, 10 Gibbs samples were collected from the model. Figure 2 shows two short fragments of articles from this NYT dataset. The background color of words indicates the probability of assigning words to the special words topic—darker colors are associated with higher probability that over the 10 Gibbs samples a word was assigned to the special topic. The words with gray foreground colors were treated as stopwords and were not included in the analysis. Figure 2(a) shows how intentionally misspelled words such as "biznesmen" and "beeznessmen" and rare

| Collection | # of Docs | Total # of Word Tokens | Median Doc Length | Mean Doc Length | # of Queries |
|---|---|---|---|---|---|
| NIPS | 1740 | 2,301,375 | 1310 | 1322.6 | N/A |
| PATENTS | 6711 | 15,014,099 | 1858 | 2237.2 | N/A |
| AP | 10000 | 2,426,181 | 235.5 | 242.6 | 142 |
| FR | 2500 | 6,332,681 | 516 | 2533.1 | 30 |

Table 1: General characteristics of document data sets used in experiments.

| NIPS | | PATENTS | | AP | | FR | |
|---|---|---|---|---|---|---|---|
| set | .0206 | fig | .0647 | tagnum | .0416 | nation | .0147 |
| number | .0167 | end | .0372 | itag | .0412 | sai | .0129 |
| results | .0153 | extend | .0267 | requir | .0381 | presid | .0118 |
| case | .0123 | invent | .0246 | includ | .0207 | polici | .0108 |
| problem | .0118 | view | .0214 | section | .0189 | issu | .0096 |
| function | .0108 | shown | .0191 | determin | .0134 | call | .0094 |
| values | .0102 | claim | .0189 | part | .0112 | support | .0085 |
| paper | .0088 | side | .0177 | inform | .0105 | need | .0079 |
| approach | .0080 | posit | .0153 | addit | .0096 | govern | .0070 |
| large | .0079 | form | .0128 | applic | .0086 | effort | .0068 |

Figure 3: Examples of background distributions (10 most likely words) learned by the SWB model for 4 different document corpora.

words such as "pinkos" are likely to be assigned to the special words topic. Figure 2(b) shows how a last name such as "Snow" and the corporation name "CSX" that are specific to the document are likely to be assigned to the special topic. The words "Snow" and "CSX" do not occur often in other documents but are mentioned several times in the example document. This combination of low document-frequency and high term-frequency within the document is one factor that makes these words more likely to be treated as "special" words.

## 4 Experimental Results: Perplexity and Precision

We use 4 different document sets in our experiments, as summarized in Table 1. The NIPS and PATENTS document sets are used for perplexity experiments and the AP and FR data sets for retrieval experiments. The NIPS data set is available online[1] and PATENTS, AP, and FR consist of documents from the U.S. Patents collection (TREC Vol-3), Associated Press news articles from 1998 (TREC Vol-2), and articles from the Federal Register (TREC Vol-1, 2) respectively. To create the sampled AP and FR data sets, all documents relevant to queries were included first and the rest of the documents were chosen randomly. In the results below all LDA/SWB/SW models were fit using $T = 200$ topics.

Figure 3 demonstrates the background component learned by the SWB model on the 4 different document data sets. The background distributions learned for each set of documents are quite intuitive, with words that are commonly used across a broad range of documents within each corpus. The ratio of words assigned to the special words distribution and the background distribution are (respectively for each data set), 25%:10% (NIPS), 58%:5% (PATENTS), 11%:6% (AP), 50%:11% (FR). Of note is the fact that a much larger fraction of words are treated as special in collections containing long documents (NIPS, PATENTS, and FR) than in short "abstract-like" collections (such as AP)—this makes sense since short documents are more likely to contain general summary information while longer documents will have more specific details.

### 4.1 Perplexity Comparisons

The NIPS and PATENTS documents sets do not have queries and relevance judgments, but nonetheless are useful for evaluating perplexity. We compare the predictive performance of the SW and SWB topic models with the standard topic model by computing the perplexity of unseen words in test documents. Perplexity of a test set under a model is defined as follows:

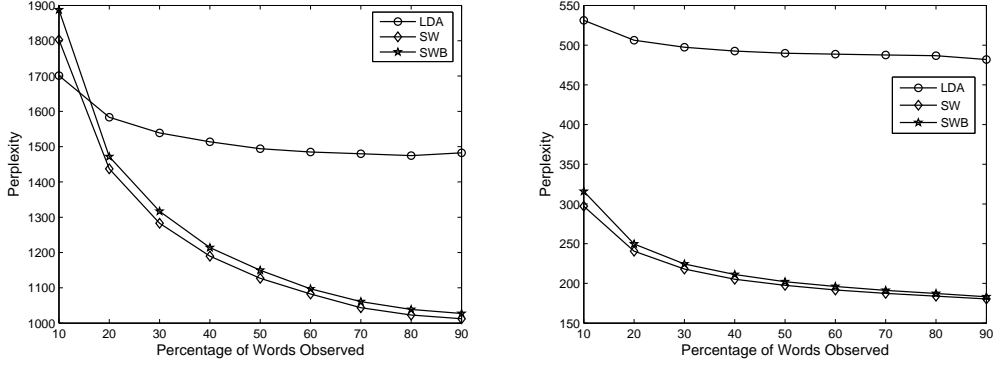

Figure 4: Average perplexity of the two special words models and the standard topics model as a function of the percentage of words observed in test documents on the NIPS data set (left) and the PATENTS data set (right).

$$\text{Perplexity}(\mathbf{w}_{test}|\mathcal{D}^{\text{train}}) = \exp\left(-\frac{\sum_{d=1}^{D_{test}} \log p(\mathbf{w}_d|\mathcal{D}^{\text{train}})}{\sum_{d=1}^{D_{test}} N_d}\right)$$

where $\mathbf{w}_{test}$ is a vector of words in the test data set, $\mathbf{w}_d$ is a vector of words in document $d$ of the test set, and $\mathcal{D}^{\text{train}}$ is the training set. For the SWB model, we approximate $p(\mathbf{w}_d|\mathcal{D}^{\text{train}})$ as follows:

$$p(\mathbf{w}_d|\mathcal{D}^{\text{train}}) \approx \frac{1}{S} \sum_{s=1}^{S} p(\mathbf{w}_d|\{\Theta^s \Phi^s \ \Psi^s \ \Omega^s \ \lambda^s\})$$

where $\Theta^s$, $\Phi^s$, $\Psi^s$, $\Omega^s$ and $\lambda^s$ are point estimates from $s = 1\!:\!S$ different Gibbs sampling runs.

The probability of the words $\mathbf{w}_d$ in a test document $d$, given its parameters, can be computed as follows:

$$p(\mathbf{w}_d|\{\Theta^s\Phi^s \ \Psi^s \ \Omega^s \ \lambda^s\}) = \prod_{i=1}^{N_d}\left[\lambda_{1d}^s \sum_{t=1}^{T} \phi_{w_it}^s \theta_{td}^s + \lambda_{2d}^s \psi_{w_id}^s + \lambda_{3d}^s \Omega_{w_i}^s\right]$$

where $N_d$ is the number of words in test document $d$ and $w_i$ is the $i$th word being predicted in the test document. $\theta_{td}^s$, $\phi_{w_it}^s$, $\psi_{w_id}^s$, $\Omega_{w_i}^s$ and $\lambda_d^s$ are point estimates from sample $s$.

When a fraction of words of a test document $d$ is observed, a Gibbs sampler is run on the observed words to update the document-specific parameters, $\theta_d$, $\psi_d$ and $\lambda_d$ and these updated parameters are used in the computation of perplexity. For the NIPS data set, documents from the last year of the data set were held out to compute perplexity ($D_{test} = 150$), and for the PATENTS data set 500 documents were randomly selected as test documents.

From the perplexity figures, it can be seen that once a small fraction of the test document words is observed (20% for NIPS and 10% for PATENTS), the SW and SWB models have significantly lower perplexity values than LDA indicating that the SW and SWB models are using the special words "route" to better learn predictive models for individual documents.

## 4.2  Information Retrieval Results

Returning to the point of capturing both specific and general aspects of documents as discussed in the introduction of the paper, we generated 500 queries of length 3-5 using randomly selected low-frequency words from the NIPS corpus and then ranked documents relative to these queries using several different methods. Table 2 shows for the top $k$-ranked documents ($k = 1, 10, 50, 100$) how many of the retrieved documents contained at least one of the words in the query. Note that we are not assessing relevance here in a traditional information retrieval sense, but instead are assessing how

| Method | 1 Ret Doc | 10 Ret Docs | 50 Ret Docs | 100 Ret Docs |
|--------|-----------|-------------|-------------|--------------|
| TF-IDF | 100.0 | 100.0 | 100.0 | 100.0 |
| LSI | 97.6 | 82.7 | 64.6 | 54.3 |
| LDA | 90.0 | 80.6 | 67.0 | 58.7 |
| SW | 99.2 | 97.1 | 79.1 | 67.3 |
| SWB | 99.4 | 96.6 | 78.7 | 67.2 |

Table 2: Percentage of retrieved documents containing at least one query word (NIPS corpus).

AP

| | MAP | | | | Pr@10d | | |
|--------|-------|------|----------|--------|-------|------|----------|
| Method | Title | Desc | Concepts | Method | Title | Desc | Concepts |
| TF-IDF | .353 | .358 | .498 | TF-IDF | .406 | .434 | .549 |
| LSI | .286 | .387 | .459 | LSI | .455 | .469 | .523 |
| LDA | .424 | .394 | .498 | LDA | .478 | .463 | .556 |
| SW | **.466*** | **.430*** | **.550*** | SW | **.524*** | **.509*** | .599* |
| SWB | .460* | .417 | .549* | SWB | .513* | .495 | **.603*** |

FR

| | MAP | | | | Pr@10d | | |
|--------|-------|------|----------|--------|-------|------|----------|
| Method | Title | Desc | Concepts | Method | Title | Desc | Concepts |
| TF-IDF | .268 | .272 | .391 | TF-IDF | .300 | .287 | .483 |
| LSI | .329 | .295 | .399 | LSI | .366 | .327 | .487 |
| LDA | .344 | .271 | .396 | LDA | .428 | .340 | .487 |
| SW | .371 | .323* | **.448*** | SW | **.469** | .407* | **.550*** |
| SWB | **.373** | **.328*** | .435 | SWB | .462 | **.423*** | .523 |

*=sig difference wrt LDA

Figure 5: Information retrieval experimental results.

often specific query words occur in retrieved documents. TF-IDF has 100% matches, as one would expect, and the techniques that generalize (such as LSI and LDA) have far fewer exact matches. The SWB and SW models have more specific matches than either LDA or LSI, indicating that they have the ability to match at the level of specific words. Of course this is not of much utility unless the SWB and SW models can also perform well in terms of retrieving relevant documents (not just documents containing the query words), which we investigate next.

For the AP and FR documents sets, 3 types of query sets were constructed from TREC Topics 1-150, based on the $Title$ (short), $Desc$ (sentence-length) and $Concepts$ (long list of keywords) fields. Queries that have no relevance judgments for a collection were removed from the query set for that collection.

The score for a document $d$ relative to a query $q$ for the SW and standard topic models can be computed as the probability of $q$ given $d$ (known as the query-likelihood model in the IR community). For the SWB topic model, we have

$$ p(q|d) \approx \prod_{w \in q} [p(x=0|d) \sum_{t=1}^{T} p(w|z=t)p(z=t|d) \; + \; p(x=1|d)p'(w|d) \; + \; p(x=2|d)p''(w)] $$

We compare SW and SWB models with the standard topic model (LDA), LSI and TF-IDF. The TF-IDF score for a word $w$ in a document $d$ is computed as TF-IDF$(w,d) = \frac{\mathbf{C}_{wd}^{WD}}{N_d} \times \log_2 \frac{D}{D_w}$. For LSI, the TF-IDF weight matrix is reduced to a $K$-dimensional latent space using SVD, $K = 200$. A given query is first mapped into the LSI latent space or the TF-IDF space (known as query folding), and documents are scored based on their cosine distances to the mapped queries.

To measure the performance of each algorithm we used 2 metrics that are widely used in IR research: the mean average precision (MAP) and the precision for the top 10 documents retrieved (pr@10d). The main difference between the AP and FR documents is that the latter documents are considerably longer on average and there are fewer queries for the FR data set. Figure 5 summarizes the results, broken down by algorithm, query type, document set, and metric. The maximum score for each query experiment is shown in bold: in all cases (query-type/data set/metric) the SW or SWB model produced the highest scores.

To determine statistical significance, we performed a t-test at the 0.05 level between the scores of each of the SW and SWB models, and the scores of the LDA model (as LDA has the best scores overall among TF-IDF, LSI and LDA). Differences between SW and SWB are not significant. In figure 5, we use the symbol * to indicate scores where the SW and SWB models showed a statistically significant difference (always an improvement) relative to the LDA model. The differences for the "non-starred" query and metric scores of SW and SWB are not statistically significant but nonetheless always favor SW and SWB over LDA.

## 5 Discussion and Conclusions

Wei and Croft (2006) have recently proposed an ad hoc LDA approach that models $p(q|d)$ as a weighted combination of a multinomial over the entire corpus (the background model), a multinomial over the document, and an LDA model. Wei and Croft showed that this combination provides excellent retrieval performance compared to other state-of-the-art IR methods. In a number of experiments (not shown) comparing the SWB and ad hoc LDA models we found that the two techniques produced comparable precision performance, with small but systematic performance gains being achieved by an ad hoc combination where the standard LDA model in ad hoc LDA was replaced with the SWB model. An interesting direction for future work is to investigate fully generative models that can achieve the performance of ad hoc approaches.

In conclusion, we have proposed a new probabilistic model that accounts for both general and specific aspects of documents or individual behavior. The model extends existing latent variable probabilistic approaches such as LDA by allowing these models to take into account specific aspects of documents (or individuals) that are exceptions to the broader structure of the data. This allows, for example, documents to be modeled as a mixture of words generated by general topics and words generated in a manner specific to that document. Experimental results on information retrieval tasks indicate that the SWB topic model does not suffer from the weakness of techniques such as LSI and LDA when faced with very specific query words, nor does it suffer the limitations of TF-IDF in terms of its ability to generalize.

**Acknowledgements**

We thank Tom Griffiths for useful initial discussions about the special words model. This material is based upon work supported by the National Science Foundation under grant IIS-0083489. We acknowledge use of the computer clusters supported by NIH grant LM-07443-01 and NSF grant EIA-0321390 to Pierre Baldi and the Institute of Genomics and Bioinformatics.

**References**

Blei, D. M., Ng, A. Y., and Jordan, M. I. (2003) Latent Dirichlet allocation, *Journal of Machine Learning Research* **3**: 993-1022.

Buntine, W., Löfström, J., Perttu, S. and Valtonen, K. (2005) Topic-specific scoring of documents for relevant retrieval *Workshop on Learning in Web Search: 22nd International Conference on Machine Learning*, pp. 34-41. Bonn, Germany.

Canny, J. (2004) GaP: a factor model for discrete data. *Proceedings of the 27th Annual SIGIR Conference*, pp. 122-129.

Daumé III, H., and Marcu, D. (2006) Domain Adaptation for Statistical classifiers. *Journal of the Artificial Intelligence Research*, **26**: 101-126.

Deerwester, S., Dumais, S. T., Furnas, G. W., Landauer, T. K., and Harshman, R. (1990) Indexing by latent semantic analysis. *Journal of the American Society for Information Science*, **41**(6): 391-407.

Griffiths, T. L., and Steyvers, M. (2004) Finding scientific topics, *Proceedings of the National Academy of Sciences*, pp. 5228-5235.

Hofmann, T. (1999) Probabilistic latent semantic indexing, *Proceedings of the 22nd Annual SIGIR Conference*, pp. 50-57.

Vogt, C. and Cottrell, G. (1999) Fusion via a linear combination of scores. *Information Retrieval*, **1**(3): 151-173.

Wei, X. and Croft, W.B. (2006) LDA-based document models for ad-hoc retrieval, *Proceedings of the 29th SIGIR Conference*, pp. 178-185.

## Footnotes

[1] From http://www.cs.toronto.edu/~roweis/data.html
